# Visual Grammars and their Neural Nets

**Eric Mjolsness**
Department of Computer Science
Yale University
New Haven, CT 06520-2158

## Abstract

I exhibit a systematic way to derive neural nets for vision problems. It involves formulating a vision problem as Bayesian inference or decision on a comprehensive model of the visual domain given by a probabilistic *grammar*.

## 1  INTRODUCTION

I show how systematically to derive optimizing neural networks that represent quantitative visual models and match them to data. This involves a design methodology which starts from first principles, namely a probabilistic model of a visual domain, and proceeds via Bayesian inference to a neural network which performs a visual task. The key problem is to find probability distributions sufficiently intricate to model general visual tasks and yet tractable enough for theory. This is achieved by probabilistic and expressive *grammars* which model the image-formation process, including heterogeneous sources of noise each modelled with a grammar rule. In particular these grammars include a crucial "relabelling" rule that removes the undetectable internal labels (or indices) of detectable features and substitutes an uninformed labeling scheme used by the perceiver.

This paper is a brief summary of the contents of [Mjolsness, 1991].

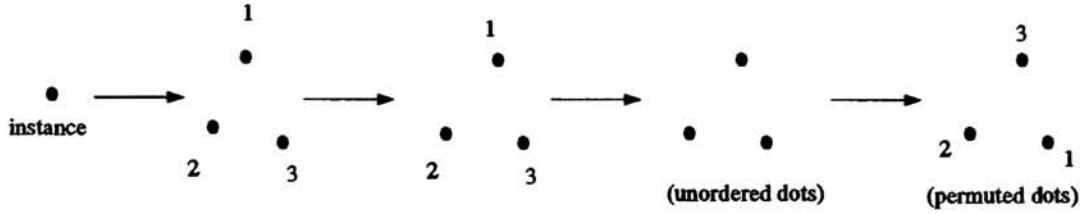

Figure 1: Operation of random dot grammar. The first arrow illustrates dot placement; the next shows dot jitter; the next arrow shows the pure, un-numbered feature locations; and the final arrow is the uninformed renumbering scheme of the perceiver.

## 2    EXAMPLE: A RANDOM-DOT GRAMMAR

The first example grammar is a generative model of pictures consisting of a number of dots (e.g. a sum of delta functions) whose relative locations are determined by one out of $M$ stored models. But the dots are subject to unknown independent jitter and an unknown global translation, and the identities of the dots (their numerical labels) are hidden from the perceiver by a random permutation operation. For example each model might represent an imaginary asterism of equally bright stars whose locations have been corrupted by instrument noise. One useful task would be to recognize which model generated the image data. The random-dot grammar is shown in (1).

| model and location | $\Gamma^0$ : | root | $\rightarrow$ | instance of model $\alpha$ at $\mathbf{x}$ |
|---|---|---|---|---|
| | | $E_0(\mathbf{x})$ | $=$ | $\frac{1}{2\sigma_r^2}|\mathbf{x}|^2$ |
| dot locations | $\Gamma^1$ : | instance$(\alpha, \mathbf{x})$ | $\rightarrow$ | $\{\text{dotloc}(\alpha, m, \hat{\mathbf{x}}_m = \mathbf{x} + \mathbf{u}_m^\alpha)\}$ |
| | | $E_1(\{\hat{\mathbf{x}}_m\})$ | $=$ | $-\log \prod_m \delta(\hat{\mathbf{x}}_m - \mathbf{x} - \mathbf{u}_m^\alpha),$ where $<\mathbf{u}_m^\alpha>_m = 0$ |
| | | | $\approx$ | $\lim_{\sigma_\delta \to 0} \frac{1}{2\sigma_\delta^2} \sum_m |\mathbf{x}_m - \mathbf{x} - \mathbf{u}_m^\alpha|^2 + c(\sigma_\delta)$ |
| dot jitter | $\Gamma^2$ : | dotloc$(\alpha, m, \hat{\mathbf{x}}_m)$ | $\rightarrow$ | dot$(m, \mathbf{x}_m)$ |
| | | $E_2(\mathbf{x}_m)$ | $=$ | $\frac{1}{2\sigma_{jit}^2}|\hat{\mathbf{x}}_m - \mathbf{x}_m|^2$ |
| scramble all dots | $\Gamma^3$ : | $\{\text{dot}(m, \mathbf{x}_m)\}$ | $\rightarrow$ | $\{\text{imagedot}(\mathbf{x}_i = \sum_m P_{m,i}\mathbf{x}_m)\}$ |
| | | $E_3(\{\mathbf{x}_i\})$ | $=$ | $-\log\left[\Pr(P)\prod_i \delta(\mathbf{x}_i - \sum_m P_{m,i}\mathbf{x}_m)\right]$ where $P$ is a permutation |

(1)

The final joint probability distribution for this grammar allows recognition and other problems to be posed as Bayesian inference and solved by neural network optimization of

$$E_{\text{final}}(\alpha, P, \mathbf{x}) = \sum_{mi} P_{m,i} \left( \frac{1}{2N\sigma_r^2} |\mathbf{x}|^2 + \frac{1}{2\sigma_{jt}^2} |\mathbf{x}_i - \mathbf{x} - \mathbf{u}_m^\alpha|^2 \right). \tag{2}$$

A sum over all permutations has been approximated by the optimization over near-permutations, as usual for Mean Field Theory networks [Yuille, 1990], resulting in a neural network implementable as an analog circuit. The fact that $P$ appears only linearly in $E_{\text{final}}$ makes the optimization problems easier; it is a generalized "assignment" problem.

## 2.1    APPROXIMATE NEURAL NETWORK WITHOUT MATCH VARIABLES

Short of approximating a $P$ configuration sum via Mean Field Theory neural nets, there is a simpler, cheaper, less accurate approximation that we have used on matching problems similar to the model recognition problem (find $\alpha$ and $\mathbf{x}$) for the dot-matching grammar. Under this approximation,

$$\text{argmax}_{\alpha, \mathbf{x}} \Pr(\alpha, \mathbf{x} | \{\mathbf{x}_i\}) \approx \text{argmax}_{\alpha, \mathbf{x}} \sum_{m,i} \exp -\frac{1}{T} \left( \frac{1}{2N\sigma_r^2} |\mathbf{x}|^2 + \frac{1}{2\sigma_{jt}^2} |\mathbf{x}_i - \mathbf{x} - \mathbf{u}_m^\alpha|^2 \right), \tag{3}$$

for $T = 1$. This objective function has a simple interpretation when $\sigma_r \to \infty$: it minimizes the Euclidean distance between two Gaussian-blurred images containing the $\mathbf{x}_i$ dots and a shifted version of the $\mathbf{u}_m$ dots respectively:

$$\begin{aligned}
&\text{argmin}_{\alpha, \mathbf{x}} \int d\mathbf{z} \, |G * I_1(\mathbf{z}) - G * I_2(\mathbf{z} - \mathbf{x})|^2 \\
= \; &\text{argmin}_{\alpha, \mathbf{x}} \int d\mathbf{z} \left| G_{\sigma/\sqrt{2}} * \sum_i \delta(\mathbf{z} - \mathbf{x}_i) - G_{\sigma/\sqrt{2}} * \sum_m \delta(\mathbf{z} - \mathbf{x} - \mathbf{u}_m^\alpha) \right|^2 \\
= \; &\text{argmin}_{\alpha, \mathbf{x}} \left[ C_1 - 2 \sum_{mi} \int d\mathbf{z} \exp -\frac{1}{\sigma^2} \left[ |\mathbf{z} - \mathbf{x}_i|^2 + |\mathbf{z} - \mathbf{x} - \mathbf{u}_m^\alpha|^2 \right] \right] \\
= \; &\text{argmax}_{\alpha, \mathbf{x}} \sum_{mi} \exp -\frac{1}{2\sigma^2} |\mathbf{x}_i - \mathbf{x} - \mathbf{u}_m^\alpha|^2
\end{aligned} \tag{4}$$

Deterministic annealing from $T = \infty$ down to $T = 1$, which is a good strategy for finding global maxima in equation (3), corresponds to a coarse-to-fine correlation matching algorithm: the global shift $\mathbf{x}$ is computed by repeated local optimization while gradually decreasing the Gaussian blur parameter $\sigma$ down to $\sigma_{jt}$.

The approximation (3) has the effect of eliminating the discrete $P_{mi}$ variables, rather than replacing them with continuous versions $V_{mi}$. The same can be said for the "elastic net" method [Durbin and Willshaw, 1987]. Compared to the elastic net, the present objective function is simpler, more symmetric between rows and columns, has a nicer interpretation in terms of known algorithms (correlation matching in scale space), and is expected to be less accurate.

# 3    EXPERIMENTS IN IMAGE REGISTRATION

Equation (3) is an objective function for recovering the global two-dimensional (2D) translation of a model consisting of arbitrarily placed dots, to match up with similar dots with jittered positions. We use it instead to find the best 2D *rotation* and horizontal translation, for two images which actually differ by a horizontal *3D* translation with roughly constant camera orientation. The images consist of *line segments* rather than single dots, some of which are *missing or extra* data. In addition, there are strong *boundary effects* due to parts of the scene being translated outside the camera's field of view. The jitter is replaced by whatever positional inaccuracies come from an actual camera producing an $128 \times 128$ image [Williams and Hanson, 1988] which is then processed by a high quality line-segment finding algorithm [Burns, 1986]. Better results would be expected of objective functions derived from grammars which explicitly model more of these noise processes, such as the grammars described in Section 4.

We experimented with minimizing this objective function with respect to unknown global translations and (sometimes) rotations, using the continuation method and sets of line segments derived from real images. The results are shown in Figures 2, 3 and 4.

# 4    MORE GRAMMARS

Going beyond the random-dot grammar, we have studied several grammars of increasing complexity. One can add rotation and dot deletion as new sources of noise, or introduce a two-level hierarchy, in which models are sets of clusters of dots. In [Mjolsness et al., 1991] we present a grammar for multiple curves in a single image, each of which is represented in the image as a set of dots that may be hard to group into their original curves. This grammar illustrates how flexible objects can be handled in our formalism.

We approach a modest plateau of generality by augmenting the hierarchical version of the random-dot grammar with multiple objects in a single scene. This degree of complexity is sufficient to introduce many interesting features of knowledge representation in high-level vision, such as multiple instances of a model in a scene, as well as requiring segmentation and grouping as part of the recognition process. We have shown [Mjolsness, 1991] that such a grammar can yield neural networks nearly identical to the "Frameville" neural networks we have previously studied as a means of mixing simple Artificial Intelligence frame systems (or semantic networks) with optimization-based neural networks. What is more, the transformation leading to Frameville is very natural. It simply pushes the permutation matrix as far back into the grammar as possible.

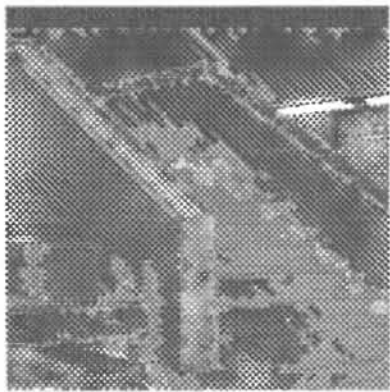
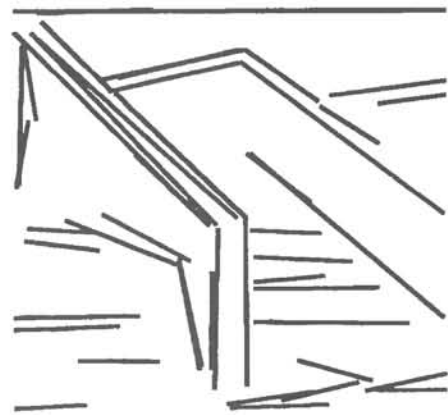

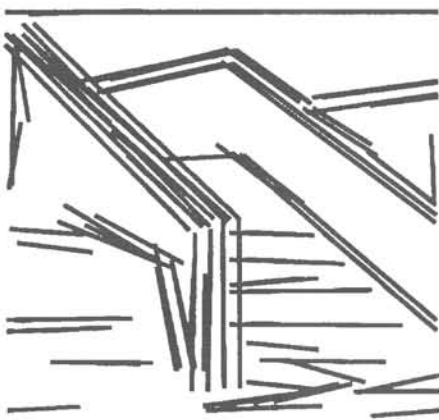
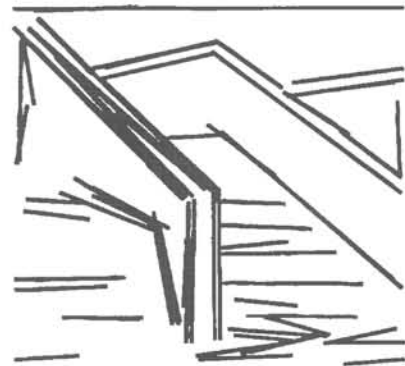

Figure 2: A simple image registration problem. (a) Stair image. (b) Long line segments derived from stair image. (c) Two unregistered line segment images derived from two images taken from two horizontally translated viewpoints in three dimensions. The images are a pair of successive frames in an image sequence. (d) Registered viersions of same data: superposed long line segments extracted from two stair images (taken from viewpoints differing by a small horizontal translation in three dimensions) that have been optimally registered in two dimensions.

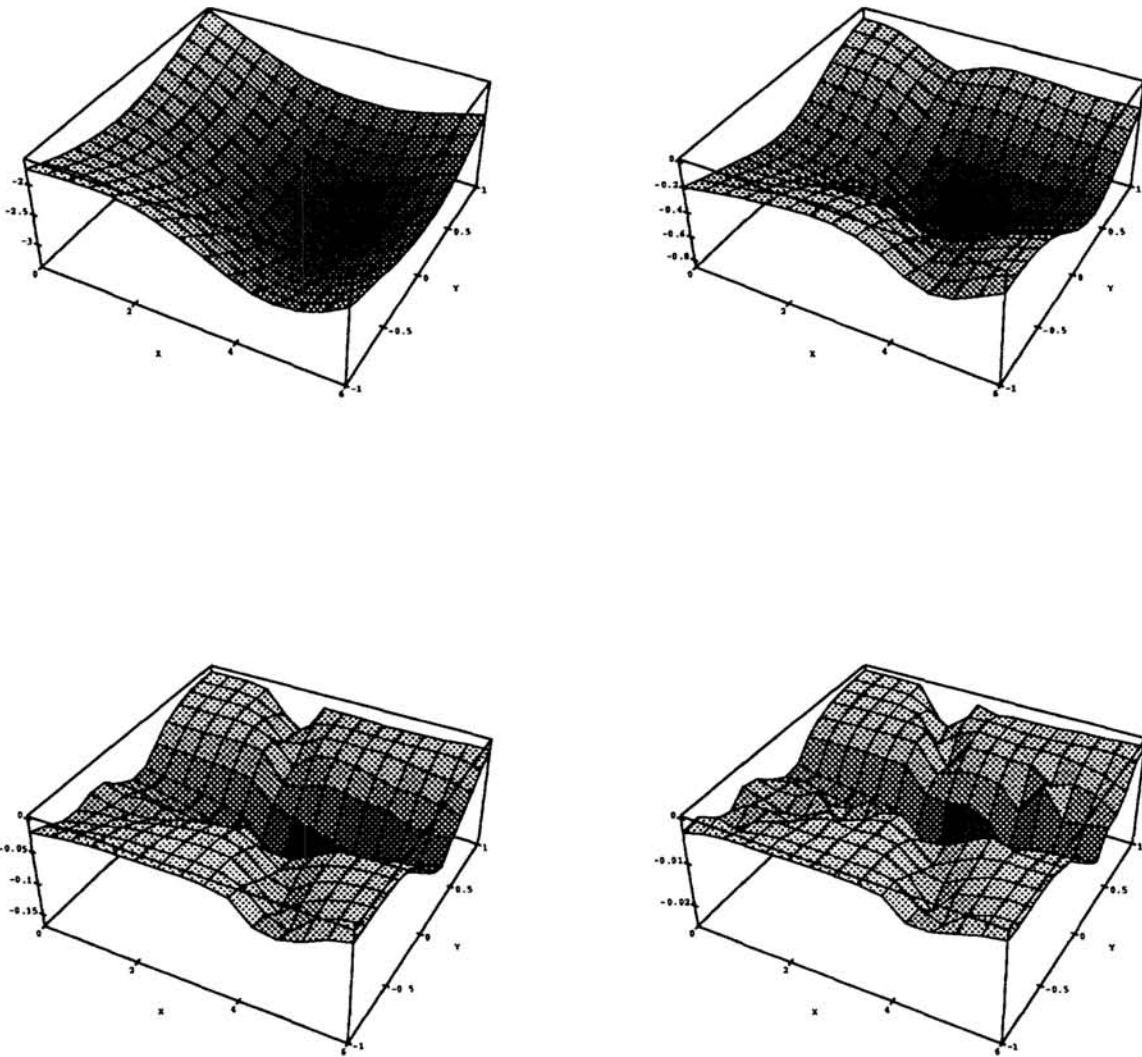

Figure 3: Continuation method (deterministic annealing). (a) Objective function at $\sigma = .0863$. (b) Objective function at $\sigma = .300$. (c) Objective function at $\sigma = .105$. (d) Objective function at $\sigma = .0364$.

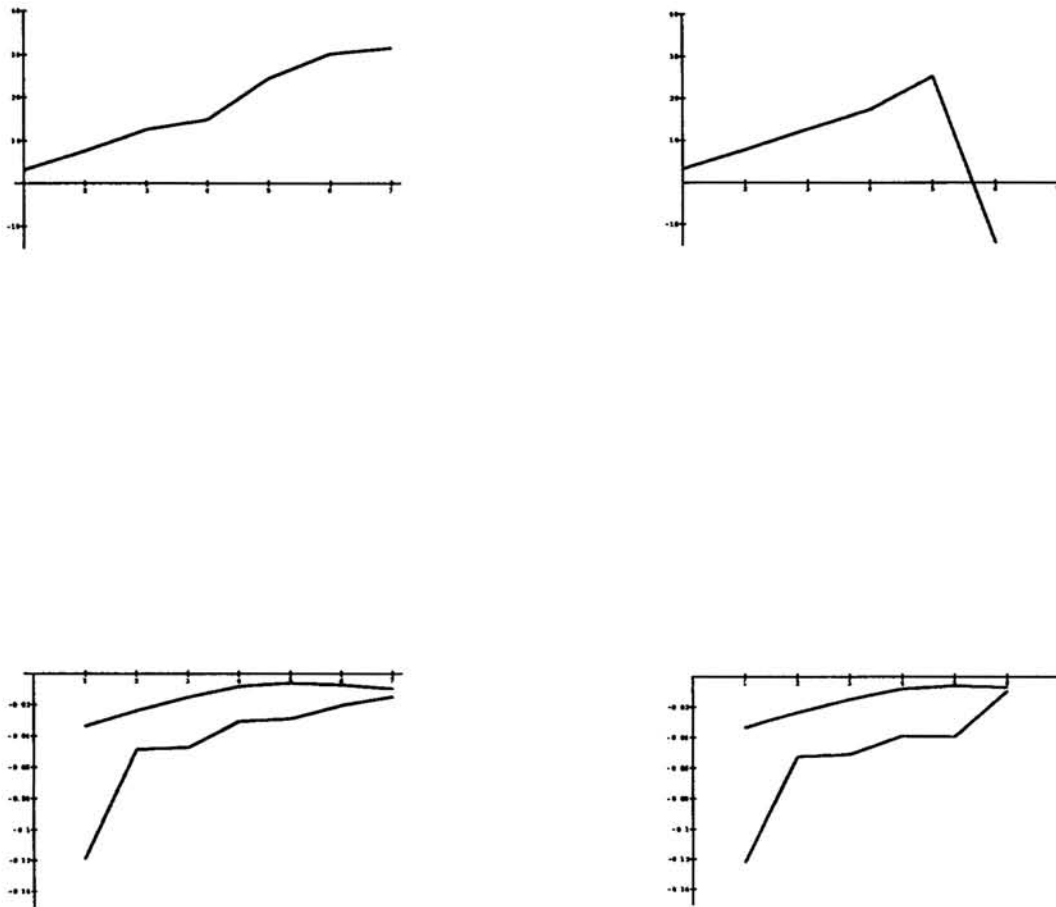

Figure 4: Image sequence displacement recovery. Frame 2 is matched to frames 3-8 in the stair image sequence. Horizontal displacements are recovered. Other starting frames yield similar results except for frame 1, which was much worse. (a) Horizontal displacement recovered, assuming no 2-d rotation. Recovered dispacement as a function of frame number is monotonic. (b) Horizontal displacement recovered, along with 2-d rotation which is found to be small except for the final frame. Displacements are in qualitative agreement with (a), more so for small displacements. (c) Objective function before and after displacement is recovered (upper and lower curves) without rotation. Note gradual decrease in $\Delta E$ with frame number (and hence with displacement). (d) Objective function before and after displacement is recovered (upper and lower curves) with rotation.

## Acknowlegements

Charles Garrett performed the computer simulations and helped formulate the line-matching objective function used therein.

## References

[Burns, 1986] Burns, J. B. (1986). Extracting straight lines. *IEEE Trans. PAMI*, 8(4):425–455.

[Durbin and Willshaw, 1987] Durbin, R. and Willshaw, D. (1987). An analog approach to the travelling salesman problem using an elastic net method. *Nature*, 326:689–691.

[Mjolsness, 1991] Mjolsness, E. (1991). Bayesian inference on visual grammars by neural nets that optimize. Technical Report YALEU/DCS/TR854, Yale University Department of Computer Science.

[Mjolsness et al., 1991] Mjolsness, E., Rangarajan, A., and Garrett, C. (1991). A neural net for reconstruction of multiple curves with a visual grammar. In *Seattle International Joint Conference on Neural Networks*.

[Williams and Hanson, 1988] Williams, L. R. and Hanson, A. R. (1988). Translating optical flow into token matches and depth from looming. In *Second International Conference on Computer Vision*, pages 441–448. Staircase test image sequence.

[Yuille, 1990] Yuille, A. L. (1990). Generalized deformable models, statistical physics, and matching problems. *Neural Computation*, 2(1):1–24.